# Learning Cue-Invariant Visual Responses

**Jarmo Hurri**
HIIT Basic Research Unit, University of Helsinki
P.O.Box 68, FIN-00014 University of Helsinki, Finland

## Abstract

Multiple visual cues are used by the visual system to analyze a scene; achromatic cues include luminance, texture, contrast and motion. Single-cell recordings have shown that the mammalian visual cortex contains neurons that respond similarly to scene structure (e.g., orientation of a boundary), regardless of the cue type conveying this information. This paper shows that cue-invariant response properties of simple- and complex-type cells can be learned from natural image data in an unsupervised manner. In order to do this, we also extend a previous conceptual model of cue invariance so that it can be applied to model simple- and complex-cell responses. Our results relate cue-invariant response properties to natural image statistics, thereby showing how the statistical modeling approach can be used to model processing beyond the elemental response properties visual neurons. This work also demonstrates how to learn, from natural image data, more sophisticated feature detectors than those based on changes in mean luminance, thereby paving the way for new data-driven approaches to image processing and computer vision.

## 1 Introduction

When segmenting a visual scene, the brain utilizes a variety of visual cues. Spatiotemporal variations in the mean luminance level – which are also called *first-order* cues – are computationally the simplest of these; the name 'first-order' comes from the idea that a single linear filtering operation can detect these cues. Other types of visual cues include contrast, texture and motion; in general, cues related to variations in other characteristics than mean luminance are called *higher-order* (also called non-Fourier) cues; the analysis of these is thought to involve more than one level of processing/filtering. Single-cell recordings have shown that the mammalian visual cortex contains neurons that are selective to both first- and higher-order cues. For example, a neuron may exhibit similar selectivity to the orientation of a boundary, regardless of whether the boundary is a result of spatial changes in mean luminance or contrast [1]. Monkey cortical areas V1 and V2, and cat cortical areas 17 and 18, contain both simple- (orientation-, frequency- and phase-selective) and complex-type (orientation- and frequency-selective, phase-invariant) cells that exhibit such *cue-invariant* response properties [2, 1, 3, 4, 5]. Previous research has been unable to pinpoint the connectivity that gives rise to cue-invariant responses.

Recent computational modeling of the visual system has produced fundamental results relating stimulus statistics to first-order response properties of simple and complex cells (see, e.g., [6, 7, 8, 9]). The contribution of this paper is to introduce a similar, natural image

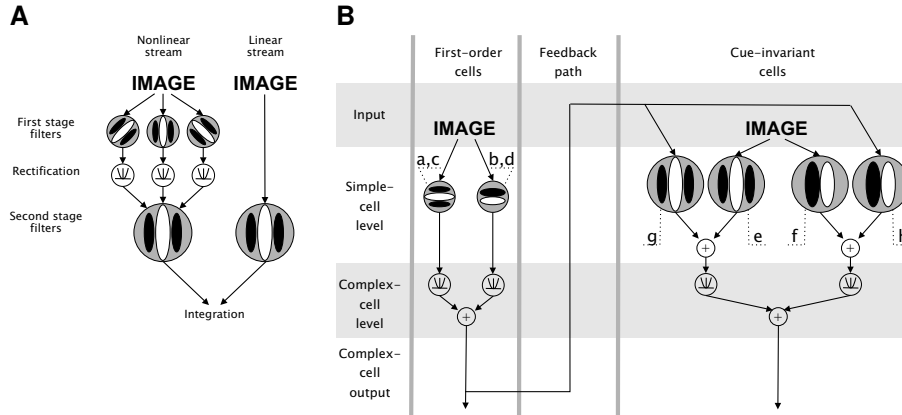

Figure 1: (**A**) The two-stream model [1], with a linear stream (on the right) and a nonlinear stream (on the left). The linear stream responds to first-order cues, while the nonlinear stream responds to higher-order cues. In the nonlinear stream, the stimulus (image) is first filtered with multiple high-frequency filters, whose outputs are transformed nonlinearly (rectified), and subsequently used as inputs for a second-stage filter. Cue-invariant responses are obtained when the outputs of these two streams are integrated. (**B**) Our model of cue-invariant responses. The model consists of simple cells, complex cells and a feedback path leading from a population of high-frequency first-order complex cells to low-frequency cue-invariant simple cells. In a cue-invariant simple cell, the feedback is filtered with a filter that has similar spatial characteristics as the feedforward filter of the cell. The output of a cue-invariant simple cell is given by the sum of the linearly filtered input and the filtered feedback. Note that while our model results in cue-invariant response properties, it is not a model of cue *integration*, because in the sum the two paths can cancel out. However, this simplification does *not* affect our results, that is, learning, since the summed output is not used in learning (see Section 3), or measurements, which excite only one of the paths significantly and do not consider integration effects (see Figures 3 and 4). In this instance of the model, the high-frequency cells prefer horizontal stimuli, while the low-frequency cue-invariant cells prefer vertical stimuli; in other instances, this relationship can be different. For actual filters used in an implementation of this model, see Figure 2. Lowercase letters a–g refer to the corresponding subfigures in Figure 2.

statistics -based framework for cue-invariant responses of both simple and complex cells. In order to achieve this, we also extend the two-stream model of cue-invariant responses (Figure 1A) to account for cue-invariant responses at both simple- and complex-cell levels.

The rest of this paper is organized as follows. In Section 2 we describe our version of the two-stream model of cue-invariant responses, which is based on feedback from complex cells to simple cells. In Section 3 we formulate an unsupervised learning rule for learning these feedback connections. We apply our learning rule to natural image data, and show that this results in the emergence of connections that give rise to cue-invariant responses at both simple- and complex-cell levels. We end this paper with conclusions in Section 4.

## 2 A model of cue-invariant responses

The most prominent model of cue-invariant responses introduced in previous research is the two-stream model (see, e.g., [1]), depicted in Figure 1A. In this research we have extended this model so that it can be applied directly to model the cue-invariant responses of simple and complex cells. Our model, shown in Figure 1B, employs standard linear-filter

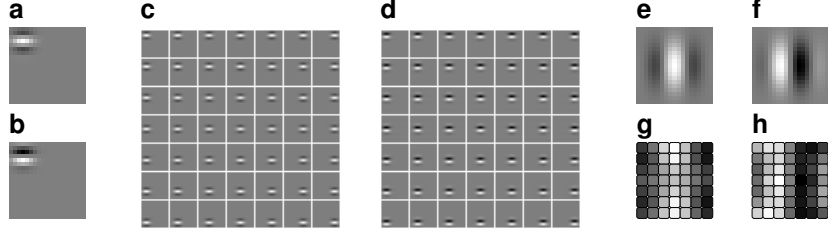

Figure 2: The filters used in an implementation of our model. The reader is referred to Figure 1B for the correspondence between subfigures (a)–(h) and the schematic model of Figure 1B. (**a**) The feedforward filter (Gabor function [10]) of a high-frequency first-order simple cell; the filter has size $19 \times 19$ pixels, which is the size of the image data in our experiments. (**b**) The feedforward filter of another first-order simple cell. This feedforward filter is otherwise similar to the one in (a), except that there is a phase difference of $\pi/2$ between the two; together, the feedforward filters in (a) and (b) are used to implement an energy model of a complex cell. (**c**) A lattice of size $7 \times 7$ of high-frequency filters of the type shown in (a); these filters are otherwise identical, except that their spatial locations vary. (**d**) A lattice of filters of the type shown in (b). Together, the lattices shown in (c) and (d) are used to implement a $7 \times 7$ lattice of energy-model complex cells with different spatial positions; the output of this lattice is the feedback relayed to the low-frequency cue-invariant cells. (**e,f**) Feedforward filters of low-frequency simple cells. (**g**) A feedback filter of size $7 \times 7$ for the simple cell whose feedforward filter is shown in (e); in order to avoid confusion between feedforward filters and feedback filters, the latter are visualized as lattices of slightly rounded rectangles. (**h**) A feedback filter for the simple cell whose feedforward filter is shown in (f). The feedback filters in (g) and (h) have been obtained by applying the learning algorithm introduced in this paper (see Section 3 for details).

models of simple cells and energy models of complex cells [10], and a feedback path from the complex-cell level to the simple-cell level. This feedback path introduces a second, nonlinear input stream to cue-invariant cells, and gives rise to cue-invariant responses in these cells. To avoid confusion between the two types of filters – one type operating on the input image and the other on the feedback – we will use the term 'feedforward filter' for the former and the term 'feedback filter' for the latter. Figure 2 shows the feedforward and feedback filters of a concrete instance (implementation) of our model. Gabor functions [10] are used to model simple-cell feedforward filters.

Figure 3 illustrates the design of higher-order gratings, and shows how the complex-cell lattice of the model transforms higher-order cues into feedback activity patterns that resemble corresponding first-order cues. A quantitative evaluation of the model is given in Figure 4. These measurements show that our model possesses the fundamental cue-invariant response properties: in our model, a cue-invariant neuron has similar selectivity to the orientation, frequency and phase of a grating stimulus, regardless of cue type (see figure caption for details). We now proceed to show how the feedback filters of our model (Figures 2g and h) can be learned from natural image data.

## 3 Learning feedback connections in an unsupervised manner

### 3.1 The objective function and the learning algorithm

In this section we introduce an unsupervised algorithm for learning feedback connection weights from complex cells to simple cells. When this learning algorithm is applied to natural image data, the resulting feedback filters are those shown in Figures 2g and h – as

| cue type | sinusoidal constituents of stimulus | | | stimulus [equation] | feedback activity |
|---|---|---|---|---|---|
| luminance | **A**  | | | **B**  <br> [=A] | **C** 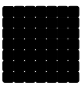 |
| texture | **D**  | **E**  | **F**  | **G**  <br> [=DE+(1-D)F] | **H**  |
| contrast | **I** 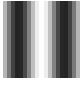 | **J**  | | **K**  <br> [=IJ] | **L**  |

Figure 3: The design of grating stimuli with different cues, and the feedback activity for these gratings. **Design of grating stimuli:** Each row illustrates how, for a particular cue, a grating stimulus is composed of sinusoidal constituents; the equation of each stimulus (B, G, K) as a function of the constituents is shown under the stimulus. Note that the orientation, frequency and phase of each grating is determined by the first sinusoidal constituent (A, D, I); here these parameters are the same for all stimuli. Here (E) and (F) are two different textures, and (I) is called the *envelope* and (J) the *carrier* of a contrast-defined stimulus. **Feedback activity:** The rightmost column shows the feedback activity – that is, response of the complex-cell lattice (see Figures 2c and d) – for the three types of stimuli. (**C**) There is no response to the luminance stimuli, since the orientation and frequency of the stimulus are different from those of the high-frequency feedforward filters. (**H, L**) For other cue types, the lattice detects the locations of energy of the vertical high-frequency constituent (E, J), thereby resulting in feedback activity that has a spatial pattern similar to a corresponding luminance pattern (A). Thus, the complex-cell lattice transforms higher-order cues into activity patterns that resemble first-order cues, and these can subsequently produce a strong response in a feedback filter (compare (H) and (L) with the feedback filter in Figure 2g). For a quantitative evaluation of the model with these stimuli, see Figure 4.

was shown in Figure 4, these feedback filters give rise to cue-invariant response properties.

The intuitive idea behind the learning algorithm is the following: in natural images, higher-order cues tend to coincide with first-order cues. For example, when two different textures are adjacent, there is often also a luminance border between them; two examples of this phenomenon are shown in Figure 5. Therefore, cue-invariant response properties could be a result of learning in which large responses in the feedforward channel (first-order responses) have become associated with large responses in the feedback channel (higher-order responses). Previous research has demonstrated the importance of such energy dependencies in modeling the visual system (see, e.g., [11, 9, 12, 13, 14]).

To turn this idea into equations, let us introduce some notation. Let vector $c(n) = [c_1(n)\, c_2(n)\, \cdots\, c_K(n)]^T$ denote the responses of a set of $K$ first-order high-frequency complex cells for the input image with index $n$. In our case the number of these complex cells is $K = 7 \times 7 = 49$ (see Figures 2c and d), so the dimension of this vector is 49. This vectorization can be done in a standard manner [15] by scanning values from the 2D lattice column-wise into a vector; when the learned feedback filter is visualized, the filter is "unvectorized" with a reverse procedure. Let $s(n)$ denote the response of a single low-

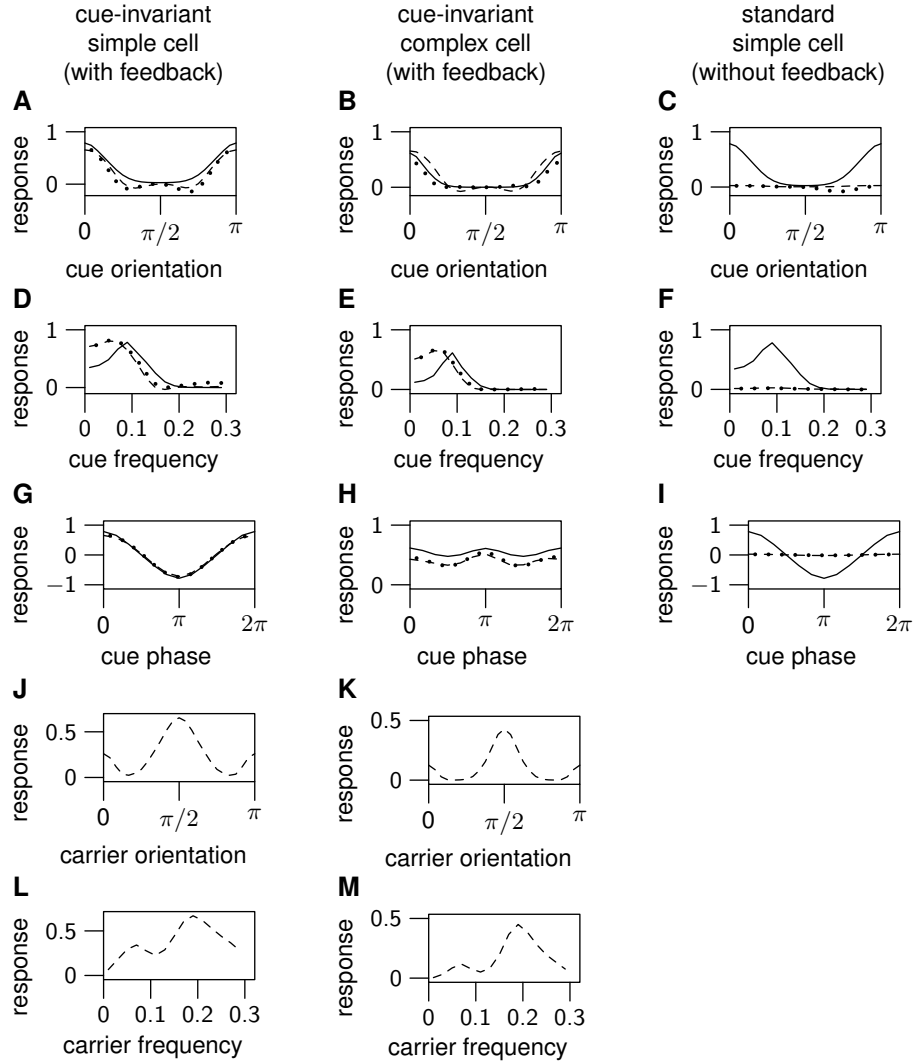

Figure 4: Our model fulfills the fundamental properties of cue-invariant responses. The plots show tuning curves for a cue-invariant simple cell – corresponding to the filters of Figures 2e and g – and complex cell of our new model (two leftmost columns), and a standard simple-cell model without feedback processing (rightmost column). Solid lines show responses to luminance-defined gratings (Figure 3B), dotted lines show responses to texture-defined gratings (Figure 3G), and dashed lines show responses to contrast-defined gratings (Figure 3K). (**A–I**) In our model, a neuron has similar selectivity to the orientation, frequency and phase of a grating stimulus, regardless of cue type; in contrast, a standard simple-cell model, without the feedback path, is only selective to the parameters of a luminance-defined grating. The preferred frequency is lower for higher-order gratings than for first-order gratings; similar observations have been made in single-cell recordings [4]. (**J–M**) In our model, the neurons are also selective to the orientation and frequency of the carrier (Figure 3J) of a contrast-defined grating (Figure 3K), thus conforming with single-cell recordings [1]. Note that these measurements were made with the feedback filters learned by our unsupervised algorithm (see Section 3); thus, these measurements confirm that learning results in cue-invariant response properties.

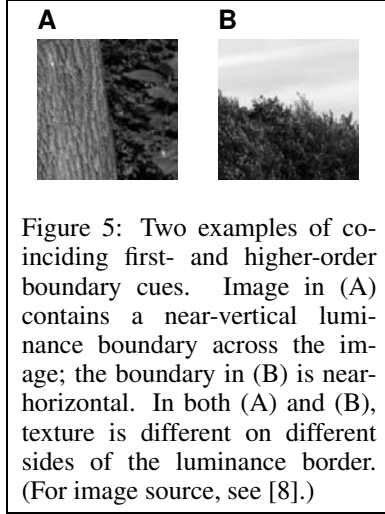

Figure 5: Two examples of co-inciding first- and higher-order boundary cues. Image in (A) contains a near-vertical luminance boundary across the image; the boundary in (B) is near-horizontal. In both (A) and (B), texture is different on different sides of the luminance border. (For image source, see [8].)

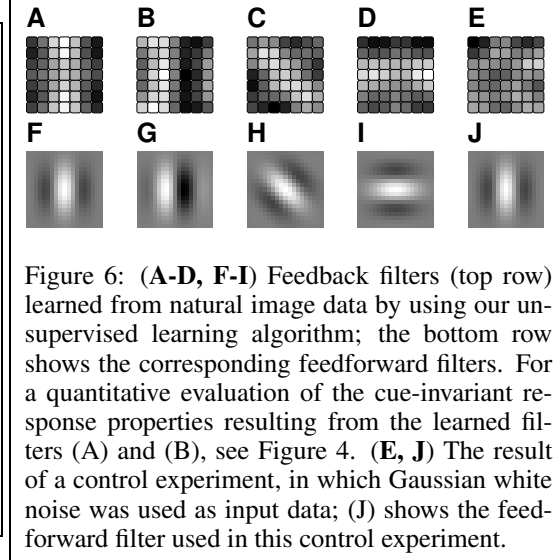

Figure 6: (**A-D, F-I**) Feedback filters (top row) learned from natural image data by using our unsupervised learning algorithm; the bottom row shows the corresponding feedforward filters. For a quantitative evaluation of the cue-invariant response properties resulting from the learned filters (A) and (B), see Figure 4. (**E, J**) The result of a control experiment, in which Gaussian white noise was used as input data; (J) shows the feedforward filter used in this control experiment.

frequency simple cell for the input image with index $n$. In our learning algorithm all the feedforward filters are fixed and only a feedback filter is learned; this means that $c(n)$ and $s(n)$ can be computed for all $n$ (all images) prior to applying the learning algorithm.

Let us denote the $K$-dimensional feedback filter with $w$; this filter is learned by our algorithm. Let $b(n) = w^T c(n)$, that is, $b(n)$ is the signal obtained when the feedback activity from the complex-cell lattice is filtered with the feedback filter; the overall activity of a cue-invariant simple cell is then $s(n) + b(n)$. Our objective function measures the correlation of energies of the feedforward response $s(n)$ and the feedback response $b(n)$:

$$f(\boldsymbol{w}) = \mathrm{E}\left\{s^2(n)b^2(n)\right\} = \boldsymbol{w}^T\mathrm{E}\left\{s^2(n)\boldsymbol{c}(n)\boldsymbol{c}(n)^T\right\}\boldsymbol{w} = \boldsymbol{w}^T\boldsymbol{M}\boldsymbol{w}, \qquad (1)$$

where $\boldsymbol{M} = \mathrm{E}\left\{s^2(n)\boldsymbol{c}(n)\boldsymbol{c}(n)^T\right\}$ is a positive-semidefinite matrix that can be computed from samples prior to learning. To keep the output of the feedback filter $b(n)$ bounded, we enforce a unit energy constraint on $b(n)$, leading into constraint

$$h(\boldsymbol{w}) = \mathrm{E}\left\{b^2(n)\right\} = \boldsymbol{w}^T\mathrm{E}\left\{\boldsymbol{c}(n)\boldsymbol{c}(n)^T\right\}\boldsymbol{w} = \boldsymbol{w}^T\boldsymbol{C}\boldsymbol{w} = 1, \qquad (2)$$

where $\boldsymbol{C} = \mathrm{E}\left\{\boldsymbol{c}(n)\boldsymbol{c}(n)^T\right\}$ is also positive-semidefinite and can be computed prior to learning. The problem of maximizing objective (1) with constraint (2) is a well-known quadratic optimization problem with a norm constraint, the solution of which is given by an eigenvalue-eigenvector problem (see below). However, in order to handle the case where $\boldsymbol{C}$ is not invertible – which will be the case below in our experiments – and to attenuate the noise in the data, we first use a technique called dimensionality reduction (see, e.g., [15]). Let $\boldsymbol{C} = \boldsymbol{E}\boldsymbol{D}\boldsymbol{E}^T$ be the eigenvalue decomposition of $\boldsymbol{C}$; in the decomposition, the eigenvectors corresponding to the $r$ smallest eigenvalues (subspaces with smallest energy; the exact value for $r$ is given in Section 3.2) have been dropped out, so $\boldsymbol{E}$ is a $K \times (K - r)$ matrix of $K - r$ eigenvectors and $\boldsymbol{D}$ is a $(K - r) \times (K - r)$ diagonal matrix containing the largest eigenvalues. Now let $\boldsymbol{v} = \boldsymbol{D}^{1/2}\boldsymbol{E}^T\boldsymbol{w}$. A one-to-one correspondence between $\boldsymbol{v}$ and $\boldsymbol{w}$ can be formed by using the pseudoinverse solution $\boldsymbol{w} = \boldsymbol{E}\boldsymbol{D}^{-1/2}\boldsymbol{v}$. Now let $\boldsymbol{z}(n) = \boldsymbol{D}^{-1/2}\boldsymbol{E}^T\boldsymbol{c}(n)$. Using these definitions of $\boldsymbol{v}$ and $\boldsymbol{z}(n)$, it is straightforward to show that the objective and constraint become $f(\boldsymbol{v}) = \boldsymbol{v}^T\mathrm{E}\left\{s^2(n)\boldsymbol{z}(n)\boldsymbol{z}(n)^T\right\}\boldsymbol{v}$ and $h(\boldsymbol{v}) = \|\boldsymbol{v}\|^2 = 1$. The global maximum $\boldsymbol{v}_{\mathrm{opt}}$ is the eigenvector of $\mathrm{E}\left\{s^2(n)\boldsymbol{z}(n)\boldsymbol{z}(n)^T\right\}$ that corresponds to the largest eigenvalue.

In practice learning from sampled data $s(n)$ and $c(n)$ proceeds as follows. First the eigenvalue decomposition of $C$ is computed. Then the transformed data set $z(n)$ is computed, and $v_{\text{opt}}$ is calculated from the eigenvalue-eigenvector problem. Finally, the optimal filter $w_{\text{opt}}$ is obtained from the pseudoinverse relationship. In learning from sampled data, all expectations are replaced with sample averages.

## 3.2 Experiments

The algorithm described above was applied to natural image data, which was sampled from a set of over 4,000 natural images [8]. The size of the sampled image patches was $19 \times 19$ pixels, and the number of samples was 250,000. The local mean (DC component) was removed from each image sample.

Simple-cell feedforward responses $s(n)$ were computed using the filter shown in Figure 2e, and the set of high-frequency complex-cell lattice activities $c(n)$ was computed using the filters shown in Figures 2c and d. A form of contrast gain control [16], which can be used to compensate for the large variation in contrast in natural images, was also applied to the natural image data: prior to filtering a natural image sample with a feedforward filter, the energy of the image was normalized inside the Gaussian modulation window of the Gabor function [10] of the feedforward filter. This preprocessing tends to weaken contrast borders, implying that in our experiments, learning higher-order responses is mostly based on texture boundaries that coincide with luminance boundaries. It should be noted, however, that in spite of this preprocessing step, the resulting feedback filters produce cue-invariant responses to both texture- and contrast-defined cues (see Figure 4). In order to make the components of $c(n)$ have zero mean, and focus on the structure of feedback activity patterns instead of overall constant activation, the local mean (DC component) was removed from each $c(n)$. To attenuate the noise in the data, the dimensionality of $c(n)$ was reduced to 16 (see Section 3.1); this retains 85% of original signal energy.

The algorithm described in Section 3.1 was then applied to this data. The resulting feedback filter is shown in Figure 6A (see also Figure 2g). Data sampling, preprocessing and the learning algorithm were then repeated, but this time using the feedforward filter shown in Figure 2f; the feedback filter obtained from this run is shown in Figure 6B (see also Figure 2h). The measurements in Figure 4 show that these feedback filters result in cue-invariant response properties at both simple- and complex-cell levels. Thus, our unsupervised algorithm learns cue-invariant response properties from natural image data. The results shown in Figures 6C and D were obtained with feedforward filters whose orientation was different from vertical, demonstrating that the observed phenomenon applies to other orientations also (in these experiments, the orientation of the high-frequency filters was orthogonal to that of the low-frequency feedforward filter).

To make sure that the results shown in Figures 6A–D are not a side effect of the preprocessing or the structure of our model, but truly reflect the statistical properties of natural image data, we ran a control experiment by repeating our first experiment, but using Gaussian white noise as input data (instead of natural image data). All other steps, including preprocessing and dimensionality reduction, were the same as in the original experiment. The result is shown in Figure 6E; as can be seen, the resulting filter lacks any spatial structure. This verifies that our original results do reflect the statistics of natural image data.

## 4 Conclusions

This paper has shown that cue-invariant response properties can be learned from natural image data in an unsupervised manner. The results were based on a model in which there is a feedback path from complex cells to simple cells, and an unsupervised algorithm which maximizes the correlation of the energies of the feedforward and filtered feedback signals.

The intuitive idea behind the algorithm is that in natural visual stimuli, higher-order cues tend to coincide with first-order cues. Simulations were performed to validate that the learned feedback filters give rise to in cue-invariant response properties.

Our results are important for three reasons. First, for the first time it has been shown that cue-invariant response properties of simple and complex cells emerge from the statistical properties of natural images. Second, our results suggest that cue invariance can result from feedback from complex cells to simple cells; no feedback from higher cortical areas would thus be needed. Third, our research demonstrates how higher-order feature detectors can be learned from natural data in an unsupervised manner; this is an important step towards general-purpose data-driven approaches to image processing and computer vision.

### Acknowledgments

The author thanks Aapo Hyvärinen and Patrik Hoyer for their valuable comments. This research was supported by the Academy of Finland (project #205742).

## References

[1] I. Mareschal and C. Baker, Jr. A cortical locus for the processing of contrast-defined contours. *Nature Neuroscience* 1(2):150–154, 1998.

[2] Y.-X. Zhou and C. Baker, Jr. A processsing stream in mammalian visual cortex neurons for non-Fourier responses. *Science* 261(5117):98–101, 1993.

[3] A. G. Leventhal, Y. Wang, M. T. Schmolesky, and Y. Zhou. Neural correlates of boundary perception. *Visual Neuroscience* 15(6):1107–1118, 1998.

[4] I. Mareschal and C. Baker, Jr. Temporal and spatial response to second-order stimuli in cat area 18. *Journal of Neurophysiology* 80(6):2811–2823, 1998.

[5] J. A. Bourne, R. Tweedale, and M. G. P. Rosa. Physiological responses of New World monkey V1 neurons to stimuli defined by coherent motion. *Cerebral Cortex* 12(11):1132–1145, 2002.

[6] B. A. Olshausen and D. Field. Emergence of simple-cell receptive field properties by learning a sparse code for natural images. *Nature* 381(6583):607–609, 1996.

[7] A. Bell and T. J. Sejnowski. The independent components of natural scenes are edge filters. *Vision Research* 37(23):3327–3338, 1997.

[8] J. H. van Hateren and A. van der Schaaf. Independent component filters of natural images compared with simple cells in primary visual cortex. *Proceedings of the Royal Society of London B* 265(1394):359–366, 1998.

[9] A. Hyvärinen and P. O. Hoyer. A two-layer sparse coding model learns simple and complex cell receptive fields and topography from natural images. *Vision Research* 41(18):2413–2423, 2001.

[10] P. Dayan and L. F. Abbott. *Theoretical Neuroscience*. The MIT Press, 2001.

[11] O. Schwartz and E. P. Simoncelli. Natural signal statistics and sensory gain control. *Nature Neuroscience* 4(8):819–825, 2001.

[12] J. Hurri and A. Hyvärinen. Simple-cell-like receptive fields maximize temporal coherence in natural video. *Neural Computation* 15(3):663–691, 2003.

[13] J. Hurri and A. Hyvärinen. Temporal and spatiotemporal coherence in simple-cell responses: a generative model of natural image sequences. *Network: Computation in Neural Systems* 14(3):527–551, 2003.

[14] Y. Karklin and M. S. Lewicki. Higher-order structure of natural images. *Network: Computation in Neural Systems* 14(3):483–499, 2003.

[15] A. Hyvärinen, J. Karhunen, and E. Oja. *Independent Component Analysis*. John Wiley & Sons, 2001.

[16] D. J. Heeger. Normalization of cell responses in cat striate cortex. *Visual Neuroscience* 9(2):181–197, 1992.
